# A Generative Model
# for Parts-based Object Segmentation

**S. M. Ali Eslami**
School of Informatics
University of Edinburgh
s.m.eslami@sms.ed.ac.uk

**Christopher K. I. Williams**
School of Informatics
University of Edinburgh
ckiw@inf.ed.ac.uk

## Abstract

The Shape Boltzmann Machine (SBM) [1] has recently been introduced as a state-of-the-art model of foreground/background object shape. We extend the SBM to account for the foreground object's *parts*. Our new model, the Multinomial SBM (MSBM), can capture both local and global statistics of part shapes accurately. We combine the MSBM with an appearance model to form a fully generative model of images of objects. Parts-based object segmentations are obtained simply by performing probabilistic inference in the model. We apply the model to two challenging datasets which exhibit significant shape and appearance variability, and find that it obtains results that are comparable to the state-of-the-art.

There has been significant focus in computer vision on object recognition and detection *e.g.* [2], but a strong desire remains to obtain richer descriptions of objects than just their bounding boxes. One such description is a *parts-based object segmentation*, in which an image is partitioned into multiple sets of pixels, each belonging to either a part of the object of interest, or its background.

The significance of parts in computer vision has been recognized since the earliest days of the field (*e.g.* [3, 4, 5]), and there exists a rich history of work on *probabilistic* models for parts-based segmentation *e.g.* [6, 7]. Many such models only consider local neighborhood statistics, however several models have recently been proposed that aim to increase the accuracy of segmentations by also incorporating prior knowledge about the foreground object's shape [8, 9, 10, 11]. In such cases, probabilistic techniques often mainly differ in how accurately they represent and learn about the variability exhibited by the shapes of the object's parts.

Accurate models of the shapes and appearances of parts can be necessary to perform inference in datasets that exhibit large amounts of variability. In general, the stronger the models of these two components, the more performance is improved. A *generative* model has the added benefit of being able to generate samples, which allows us to visually inspect the quality of its understanding of the data and the problem.

Recently, a generative probabilistic model known as the Shape Boltzmann Machine (SBM) has been used to model *binary* object shapes [1]. The SBM has been shown to constitute the state-of-the-art and it possesses several highly desirable characteristics: samples from the model look realistic, and it generalizes to generate samples that differ from the limited number of examples it is trained on.

The main contributions of this paper are as follows: 1) In order to account for object parts we extend the SBM to use multinomial visible units instead of binary ones, resulting in the Multinomial Shape Boltzmann Machine (MSBM), and we demonstrate that the MSBM constitutes a strong model of parts-based object shape. 2) We combine the MSBM with an appearance model to form a fully generative model of images of objects (see Fig. 1). We show how parts-based object segmentations can be obtained simply by performing probabilistic inference in the model. We apply our model to two challenging datasets and find that in addition to being principled and fully generative, the model's performance is comparable to the state-of-the-art.

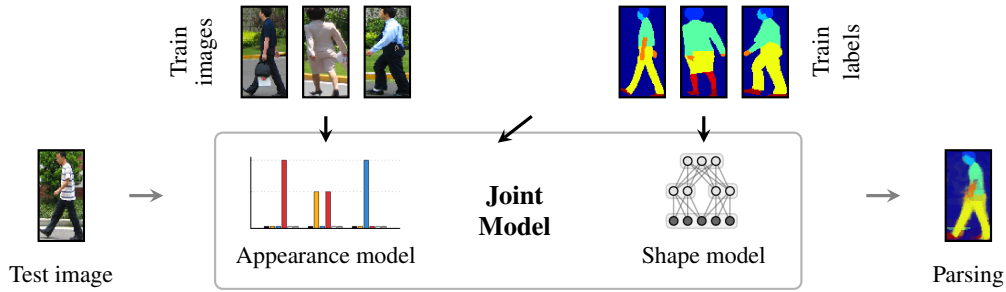

Figure 1: **Overview.** Using annotated images separate models of shape and appearance are trained. Given an unseen test image, its parsing is obtained via inference in the proposed joint model.

In Secs. 1 and 2 we present the model and propose efficient inference and learning schemes. In Sec. 3 we compare and contrast the resulting joint model with existing work in the literature. We describe our experimental results in Sec. 4 and conclude with a discussion in Sec. 5.

# 1 Model

We consider datasets of cropped images of an object class. We assume that the images are constructed through some combination of a fixed number of parts. Given a dataset $\mathbf{D} = \{\mathbf{X}^d\}, d = 1...n$ of such images $\mathbf{X}$, each consisting of $P$ pixels $\{\mathbf{x}_i\}, i = 1...P$, we wish to infer a segmentation $\mathbf{S}$ for the image. $\mathbf{S}$ consists of a labeling $\mathbf{s}_i$ for every pixel, where $\mathbf{s}_i$ is a 1-of-$(L+1)$ encoded variable, and $L$ is the fixed number of parts that combine to generate the foreground. In other words, $\mathbf{s}_i = (s_{li})$, $l = 0...L, s_{li} \in \{0, 1\}$ and $\sum_l s_{li} = 1$. Note that the background is also treated as a 'part' ($l = 0$). Accurate inference of $\mathbf{S}$ is driven by models for 1) part shapes and 2) part appearances.

**Part shapes:** Several types of models can be used to define probabilistic distributions over segmentations $\mathbf{S}$. The simplest approach is to model each pixel $\mathbf{s}_i$ independently with categorical variables whose parameters are specified by the object's mean shape (Fig. 2(a)). Markov Random Fields (MRFs, Fig. 2(b)) additionally model interactions between nearby pixels using pairwise potential functions that efficiently capture local properties of images like smoothness and continuity.

Restricted Boltzmann Machines (RBMs) and their multi-layered counterparts Deep Boltzmann Machines (DBMs, Fig. 2(c)) make heavy use of hidden variables to efficiently define higher-order potentials that take into account the configuration of larger groups of image pixels. The introduction of such hidden variables provides a way to efficiently capture complex, global properties of image pixels. RBMs and DBMs are powerful generative models, but they also have many parameters. Segmented images, however, are expensive to obtain and datasets are typically small (hundreds of examples). In order to learn a model that accurately captures the properties of part shapes we use DBMs but also impose carefully chosen connectivity and capacity constraints, following the structure of the Shape Boltzmann Machine (SBM) [1]. We further extend the model to account for multi-part shapes to obtain the Multinomial Shape Boltzmann Machine (MSBM).

The MSBM has two layers of latent variables: $\mathbf{h}^1$ and $\mathbf{h}^2$ (collectively $\mathbf{H} = \{\mathbf{h}^1, \mathbf{h}^2\}$), and defines a Boltzmann distribution over segmentations $p(\mathbf{S}) = \sum_{\mathbf{h}^1, \mathbf{h}^2} \exp\{-E(\mathbf{S}, \mathbf{h}^1, \mathbf{h}^2 | \theta^s)\} / Z(\theta^s)$ where

$$E(\mathbf{S}, \mathbf{h}^1, \mathbf{h}^2 | \theta^s) = \sum_{i,l} b_{li} s_{li} + \sum_{i,j,l} w^1_{lij} s_{li} h^1_j + \sum_j c^1_j h^1_j + \sum_{j,k} w^2_{jk} h^1_j h^2_k + \sum_k c^2_k h^2_k, \quad (1)$$

where $j$ and $k$ range over the first and second layer hidden variables, and $\theta^s = \{W^1, W^2, \mathbf{b}, \mathbf{c}^1, \mathbf{c}^2\}$ are the shape model parameters. In the first layer, local receptive fields are enforced by connecting each hidden unit in $\mathbf{h}^1$ only to a subset of the visible units, corresponding to one of four patches, as shown in Fig. 2(d,e). Each patch overlaps its neighbor by $b$ pixels, which allows boundary continuity to be learned at the lowest layer. We share weights between the four sets of first-layer hidden units and patches, and purposely restrict the number of units in $\mathbf{h}^2$. These modifications significantly reduce the number of parameters whilst taking into account an important property of shapes, namely that the strongest dependencies between pixels are typically local.

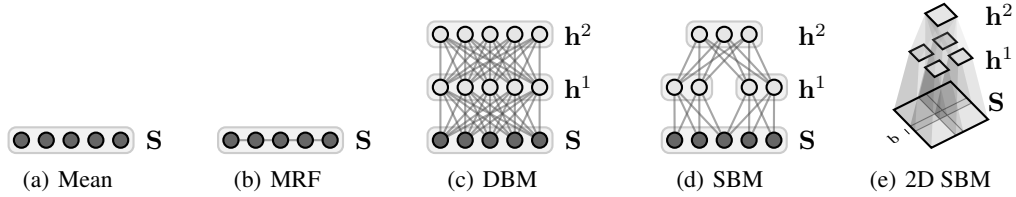

(a) Mean     (b) MRF     (c) DBM     (d) SBM     (e) 2D SBM

Figure 2: **Models of shape.** Object shape is modeled with undirected graphical models. (a) 1D slice of a mean model. (b) Markov Random Field in 1D. (c) Deep Boltzmann Machine in 1D. (d) 1D slice of a Shape Boltzmann Machine. (e) Shape Boltzmann Machine in 2D. In all models latent units **h** are binary and visible units **S** are multinomial random variables. Based on Fig. 2 of [1].

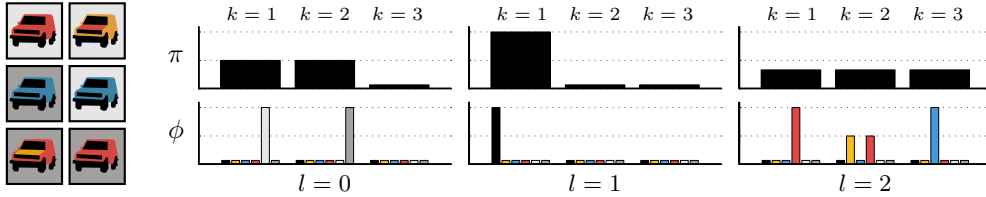

Figure 3: **A model of appearances.** *Left:* An exemplar dataset. Here we assume one background ($l = 0$) and two foreground ($l = 1$, non-body; $l = 2$, body) parts. *Right:* The corresponding appearance model. In this example, $L = 2$, $K = 3$ and $W = 6$. Best viewed in color.

**Part appearances:** Pixels in a given image are assumed to have been generated by $W$ fixed Gaussians in RGB space. During pre-training, the means $\{\mu_w\}$ and covariances $\{\Sigma_w\}$ of these Gaussians are extracted by training a mixture model with $W$ components on every pixel in the dataset, ignoring image and part structure. It is also assumed that each of the $L$ parts can have different appearances in different images, and that these appearances can be clustered into $K$ classes. The classes differ in how likely they are to use each of the $W$ components when 'coloring in' the part.

The generative process is as follows. For part $l$ in an image, one of the $K$ classes is chosen (represented by a 1-of-$K$ indicator variable $\mathbf{a}_l$). Given $\mathbf{a}_l$, the probability distribution defined on pixels associated with part $l$ is given by a Gaussian mixture model with means $\{\mu_w\}$ and covariances $\{\Sigma_w\}$ and mixing proportions $\{\phi_{lkw}\}$. The prior on $\mathbf{A} = \{\mathbf{a}_l\}$ specifies the probability $\pi_{lk}$ of appearance class $k$ being chosen for part $l$. Therefore appearance parameters $\theta^a = \{\pi_{lk}, \phi_{lkw}\}$ (see Fig. 3) and:

$$p(\mathbf{x}_i|\mathbf{A}, \mathbf{s}_i, \theta^a) = \prod_l p(\mathbf{x}_i|\mathbf{a}_l, \theta^a)^{s_{li}} = \prod_l \left( \prod_k \left( \sum_w \phi_{lkw} \mathcal{N}(\mathbf{x}_i|\mu_w, \Sigma_w) \right)^{a_{lk}} \right)^{s_{li}}, \quad (2)$$

$$p(\mathbf{A}|\theta^a) = \prod_l p(\mathbf{a}_l|\theta^a) = \prod_l \prod_k (\pi_{lk})^{a_{lk}}. \quad (3)$$

**Combining shapes and appearances:** To summarize, the latent variables for $\mathbf{X}$ are $\mathbf{A}$, $\mathbf{S}$, $\mathbf{H}$, and the model's active parameters $\theta$ include shape parameters $\theta^s$ and appearance parameters $\theta^a$, so that

$$p(\mathbf{X}, \mathbf{A}, \mathbf{S}, \mathbf{H}|\theta) = \frac{1}{Z(\lambda)} p(\mathbf{A}|\theta^a) p(\mathbf{S}, \mathbf{H}|\theta^s) \prod_i p(\mathbf{x}_i|\mathbf{A}, \mathbf{s}_i, \theta^a)^\lambda, \quad (4)$$

where the parameter $\lambda$ adjusts the relative contributions of the shape and appearance components. See Fig. 4 for an illustration of the complete graphical model. During learning, we find the values of $\theta$ that maximize the likelihood of the training data $\mathbf{D}$, and segmentation is performed on a previously-unseen image by querying the marginal distribution $p(\mathbf{S}|\mathbf{X}^{\text{test}}, \theta)$. Note that $Z(\lambda)$ is constant throughout the execution of the algorithms. We set $\lambda$ via trial and error in our experiments.

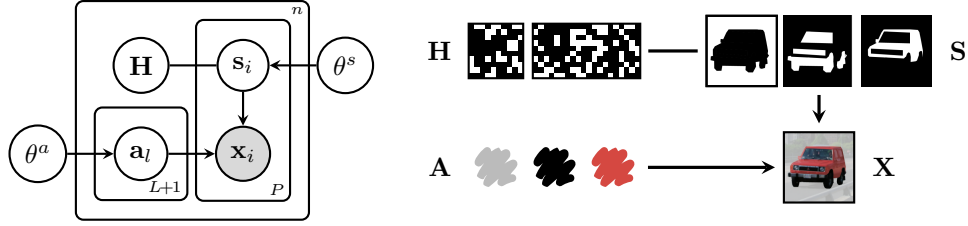

Figure 4: **A model of shape *and* appearance.** *Left:* The joint model. Pixels $\mathbf{x}_i$ are modeled via appearance variables $\mathbf{a}_l$. The model's belief about each layer's shape is captured by shape variables $\mathbf{H}$. Segmentation variables $\mathbf{s}_i$ assign each pixel to a layer. *Right:* Schematic for an image $\mathbf{X}$.

## 2 Inference and learning

**Inference:** We approximate $p(\mathbf{A}, \mathbf{S}, \mathbf{H}|\mathbf{X}, \theta)$ by drawing samples of $\mathbf{A}$, $\mathbf{S}$ and $\mathbf{H}$ using block-Gibbs Markov Chain Monte Carlo (MCMC). The desired distribution $p(\mathbf{S}|\mathbf{X}, \theta)$ can then be obtained by considering only the samples for $\mathbf{S}$ (see Algorithm 1). In order to sample $p(\mathbf{A}|\mathbf{S}, \mathbf{H}, \mathbf{X}, \theta)$ we consider the conditional distribution of appearance class $k$ being chosen for part $l$ which is given by:

$$p(a_{lk} = 1|\mathbf{S}, \mathbf{X}, \theta) = \frac{\pi_{lk} \prod_i \left( \sum_w \phi_{lkw} \mathcal{N}(\mathbf{x}_i|\mu_w, \Sigma_w) \right)^{\lambda \cdot s_{li}}}{\sum_{r=1}^{K} \left[ \pi_{lr} \prod_i \left( \sum_w \phi_{lrw} \mathcal{N}(\mathbf{x}_i|\mu_w, \Sigma_w) \right)^{\lambda \cdot s_{li}} \right]}. \tag{5}$$

Since the MSBM only has edges between each pair of adjacent layers, all hidden units within a layer are conditionally independent given the units in the other two layers. This property can be exploited to make inference in the shape model exact and efficient. The conditional probabilities are:

$$p(h_j^1 = 1|\mathbf{s}, \mathbf{h}^2, \theta) = \sigma(\sum_{i,l} w_{lij}^1 s_{li} + \sum_k w_{jk}^2 h_k^2 + c_j^1), \tag{6}$$

$$p(h_k^2 = 1|\mathbf{h}^1, \theta) = \sigma(\sum_j w_{jk}^2 h_j^1 + c_j^2), \tag{7}$$

where $\sigma(y) = 1/(1 + \exp(-y))$ is the sigmoid function. To sample from $p(\mathbf{H}|\mathbf{S}, \mathbf{X}, \theta)$ we iterate between Eqns. 6 and 7 multiple times and keep only the final values of $\mathbf{h}^1$ and $\mathbf{h}^2$. Finally, we draw samples for the pixels in $p(\mathbf{S}|\mathbf{A}, \mathbf{H}, \mathbf{X}, \theta)$ independently:

$$p(s_{li} = 1|\mathbf{A}, \mathbf{H}, \mathbf{X}, \theta) = \frac{\exp(\sum_j w_{lij}^1 h_j^1 + b_{li}) \, p(\mathbf{x}_i|\mathbf{A}, s_{li} = 1, \theta)^{\lambda}}{\sum_{m=1}^{L} \exp(\sum_j w_{mij}^1 h_j^1 + b_{mi}) \, p(\mathbf{x}_i|\mathbf{A}, s_{mi} = 1, \theta)^{\lambda}}. \tag{8}$$

**Seeding:** Since the latent-space is extremely high-dimensional, in practice we find it helpful to run several inference chains, each initializing $\mathbf{S}^{(1)}$ to a different value. The 'best' inference is retained and the others are discarded. The computation of the likelihood $p(\mathbf{X}|\theta)$ of image $\mathbf{X}$ is intractable, so we approximate the quality of each inference using a scoring function:

$$\text{Score}(\mathbf{X}|\theta) = \frac{1}{T} \sum_t p(\mathbf{X}, \mathbf{A}^{(t)}, \mathbf{S}^{(t)}, \mathbf{H}^{(t)}|\theta), \tag{9}$$

where $\{\mathbf{A}^{(t)}, \mathbf{S}^{(t)}, \mathbf{H}^{(t)}\}, t = 1...T$ are the samples obtained from the posterior $p(\mathbf{A}, \mathbf{S}, \mathbf{H}|\mathbf{X}, \theta)$. If the samples were drawn from the prior $p(\mathbf{A}, \mathbf{S}, \mathbf{H}|\theta)$ the scoring function would be an unbiased estimator of $p(\mathbf{X}|\theta)$, but would be wildly inaccurate due to the high probability of missing the important regions of latent space (see *e.g.* [12, p. 107-109] for further discussion of this issue).

**Learning:** Learning of the model involves maximizing the log likelihood $\log p(\mathbf{D}|\theta^a, \theta^s)$ of the training dataset $\mathbf{D}$ with respect to the model parameters $\theta^a$ and $\theta^s$. Since training is partially supervised, in that for each image $\mathbf{X}$ its corresponding segmentation $\mathbf{S}$ is also given, we can learn the parameters of the shape and appearance components separately.

For appearances, the learning of the mixing coefficients and the histogram parameters decomposes into standard mixture updates independently for each part. For shapes, we follow the standard deep

**Algorithm 1** MCMC inference algorithm.

1: **procedure** INFER($\mathbf{X}, \theta$)
2:      Initialize $\mathbf{S}^{(1)}, \mathbf{H}^{(1)}$
3:      **for** $t \leftarrow 2 : chain\_length$ **do**
4:          $\mathbf{A}^{(t)} \sim p(\mathbf{A}|\mathbf{S}^{(t-1)}, \mathbf{H}^{(t-1)}, \mathbf{X}, \theta)$
5:          $\mathbf{S}^{(t)} \sim p(\mathbf{S}|\mathbf{A}^{(t)}, \mathbf{H}^{(t-1)}, \mathbf{X}, \theta)$
6:          $\mathbf{H}^{(t)} \sim p(\mathbf{H}|\mathbf{S}^{(t)}, \theta)$
7:      **return** $\{\mathbf{S}^{(t)}\}_{t=burnin:chain\_length}$

learning literature closely [13, 1]. In the pre-training phase we greedily train the model bottom up, one layer at a time. We begin by training an RBM on the observed data using stochastic maximum likelihood learning (SML; also referred to as 'persistent CD'; [14, 13]). Once this RBM is trained, we infer the conditional mean of the hidden units for each training image. The resulting vectors then serve as the training data for a second RBM which is again trained using SML. We use the parameters of these two RBMs to initialize the parameters of the full MSBM model. In the second phase we perform approximate stochastic gradient ascent in the likelihood of the full model to fine-tune the parameters in an EM-like scheme as described in [13].

## 3 Related work

Existing probabilistic models of images can be categorized by the amount of variability they expect to encounter in the data and by how they model this variability. A significant portion of the literature models images using only two parts: a foreground object and its background *e.g.* [15, 16, 17, 18, 19]. Models that account for the parts within the foreground object mainly differ in how accurately they learn about and represent the variability of the shapes of the object's parts.

In Probabilistic Index Maps (PIMs) [8] a mean partitioning is learned, and the deformable PIM [9] additionally allows for local deformations of this mean partitioning. Stel Component Analysis [10] accounts for larger amounts of shape variability by learning a number of different template means for the object that are blended together on a pixel-by-pixel basis. Factored Shapes and Appearances [11] models global properties of shape using a factor analysis-like model, and 'masked' RBMs have been used to model more local properties of shape [20]. However, none of these models constitute a strong model of shape in terms of realism of samples and generalization capabilities [1]. We demonstrate in Sec. 4 that, like the SBM, the MSBM does in fact possess these properties.

The closest works to ours in terms of ability to deal with datasets that exhibit significant variability in both shape and appearance are the works of Bo and Fowlkes [21] and Thomas *et al.* [22]. Bo and Fowlkes [21] present an algorithm for pedestrian segmentation that models the shapes of the parts using several template means. The different parts are composed using hand coded geometric constraints, which means that the model cannot be automatically extended to other application domains. The Implicit Shape Model (ISM) used in [22] is reliant on interest point detectors and defines distributions over segmentations only in the *posterior*, and therefore is not fully generative. The model presented here is entirely learned from data and fully generative, therefore it can be applied to new datasets and diagnosed with relative ease. Due to its modular structure, we also expect it to rapidly absorb future developments in shape and appearance models.

## 4 Experiments

**Penn-Fudan pedestrians:** The first dataset that we considered is Penn-Fudan pedestrians [23], consisting of 169 images of pedestrians (Fig. 6(a)). The images are annotated with ground-truth segmentations for $L = 7$ different parts (hair, face, upper and lower clothes, shoes, legs, arms; Fig. 6(d)). We compare the performance of the model with the algorithm of Bo and Fowlkes [21].

For the shape component, we trained an MSBM on the 684 images of a labeled version of the HumanEva dataset [24] (at $48 \times 24$ pixels; also flipped horizontally) with overlap $b = 4$, and 400 and 50 hidden units in the first and second layers respectively. Each layer was pre-trained for 3000 epochs (iterations). After pre-training, joint training was performed for 1000 epochs.

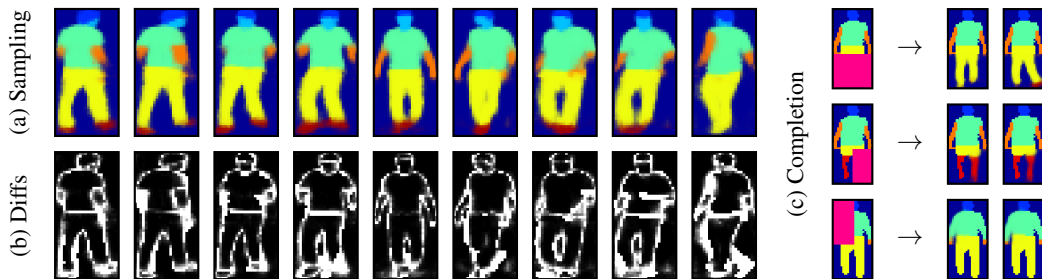

Figure 5: **Learned shape model.** (a) A chain of samples (1000 samples between frames). The apparent 'blurriness' of samples is *not* due to averaging or resizing. We display the *probability* of each pixel belonging to different parts. If, for example, there is a 50-50 chance that a pixel belongs to the red or blue parts, we display that pixel in purple. (b) Differences between the samples and their most similar counterparts in the training dataset. (c) Completion of occlusions (pink).

To assess the realism and generalization characteristics of the learned MSBM we sample from it. In Fig. 5(a) we show a chain of unconstrained samples from an MSBM generated via block-Gibbs MCMC (1000 samples between frames). The model captures highly non-linear correlations in the data whilst preserving the object's details (*e.g.* face and arms). To demonstrate that the model has not simply memorized the training data, in Fig. 5(b) we show the difference between the sampled shapes in Fig. 5(a) and their closest images in the training set (based on per-pixel label agreement). We see that the model generalizes in non-trivial ways to generate *realistic* shapes that it had not encountered during training. In Fig. 5(c) we show how the MSBM completes rectangular occlusions. The samples highlight the variability in possible completions captured by the model. Note how, *e.g.* the length of the person's trousers on one leg affects the model's predictions for the other, demonstrating the model's knowledge about long-range dependencies. An interactive MATLAB GUI for sampling from this MSBM has been included in the supplementary material.

The Penn-Fudan dataset (at $200 \times 100$ pixels) was then split into 10 train/test cross-validation splits without replacement. We used the training images in each split to train the appearance component with a vocabulary of size $W = 50$ and $K = 100$ mixture components[1]. We additionally constrained the model by sharing the appearance models for the arms and legs with that of the face.

We assess the quality of the appearance model by performing the following experiment: for each test image, we used the scoring function described in Eq. 9 to evaluate a number of different proposal segmentations for that image. We considered 10 randomly chosen segmentations from the training dataset *as well* as the ground-truth segmentation for the test image, and found that the appearance model correctly assigns the highest score to the ground-truth 95% of the time.

During inference, the shape and appearance models (which are defined on images of different sizes), were combined at $200 \times 100$ pixels via MATLAB's `imresize` function, and we set $\lambda = 0.8$ (Eq. 8) via trial and error. Inference chains were seeded at 100 exemplar segmentations from the HumanEva dataset (obtained using the $K$-medoids algorithm with $K = 100$), and were run for 20 Gibbs iterations each (with 5 iterations of Eqs. 6 and 7 per Gibbs iteration). Our unoptimized MATLAB implementation completed inference for each chain in around 7 seconds.

We compute the conditional probability of each pixel belonging to different parts given the last set of samples obtained from the highest scoring chain, assign each pixel independently to the most likely part at that pixel, and report the percentage of correctly labeled pixels (see Table 1). We find that accuracy can be improved using superpixels (SP) computed on $\mathbf{X}$ (pixels within a superpixel are all assigned the most common label within it; as with [21] we use gPb-OWT-UCM [25]). We also report the accuracy obtained, had the top scoring seed segmentation been returned as the final segmentation for each image. Here the quality of the seed is determined solely by the appearance model. We observe that the model has comparable performance to the state-of-the-art but pedestrian-specific algorithm of [21], and that inference in the model significantly improves the accuracy of the segmentations over the baseline (top seed+SP). Qualitative results can be seen in Fig. 6(c).

Table 1: **Penn-Fudan pedestrians.** We report the percentage of correctly labeled pixels. The final column is an average of the background, upper and lower body scores (as reported in [21]).

|  | FG | BG | Upper Body | Lower Body | Head | Average |
|---|---|---|---|---|---|---|
| Bo and Fowlkes [21] | 73.3% | 81.1% | 73.6% | 71.6% | 51.8% | 69.5% |
| MSBM | 70.7% | 72.8% | 68.6% | 66.7% | 53.0% | 65.3% |
| MSBM + SP | 71.6% | 73.8% | 69.9% | 68.5% | 54.1% | 66.6% |
| Top seed | 59.0% | 61.8% | 56.8% | 49.8% | 45.5% | 53.5% |
| Top seed + SP | 61.6% | 67.3% | 60.8% | 54.1% | 43.5% | 56.4% |

Table 2: **ETHZ cars.** We report the percentage of pixels belonging to each part that are labeled correctly. The final column is an average weighted by the frequency of occurrence of each label.

|  | BG | Body | Wheel | Window | Bumper | License | Light | Average |
|---|---|---|---|---|---|---|---|---|
| ISM [22] | 93.2% | 72.2% | 63.6% | 80.5% | 73.8% | 56.2% | 34.8% | 86.8% |
| MSBM | 94.6% | 72.7% | 36.8% | 74.4% | 64.9% | 17.9% | 19.9% | 86.0% |
| Top seed | 92.2% | 68.4% | 28.3% | 63.8% | 45.4% | 11.2% | 15.1% | 81.8% |

**ETHZ cars:** The second dataset that we considered is the ETHZ labeled cars dataset [22], which itself is a subset of the LabelMe dataset [23], consisting of 139 images of cars, all in the same semi-profile view (Fig. 7(a)). The images are annotated with ground-truth segmentations for $L = 6$ parts (body, wheel, window, bumper, license plate, headlight; Fig. 7(d)). We compare the performance of the model with the ISM of Thomas *et al.* [22], who also report their results on this dataset.

The dataset was split into 10 train/test cross-validation splits without replacement. We used the training images in each split to train both the shape and appearance components. For the shape component, we trained an MSBM at $50 \times 50$ pixels with overlap $b = 4$, and 2000 and 100 hidden units in the first and second layers respectively. Each layer was pre-trained for 3000 epochs and joint training was performed for 1000 epochs. The appearance model was trained with a vocabulary of size $W = 50$ and $K = 100$ mixture components and we set $\lambda = 0.7$. Inference chains were seeded at 50 exemplar segmentations (obtained using $K$-medoids). We find that the use of superpixels does not help with this dataset (due to the poor quality of superpixels obtained for these images).

Qualitative and quantitative results that show the performance of model to be comparable to the state-of-the-art ISM can be seen in Fig. 7(c) and Table 2. We believe the discrepancy in accuracy between the MSBM and ISM on the 'license' and 'light' labels to mainly be due to ISM's use of interest-points, as they are able to locate such fine structures accurately. By incorporating better models of part appearance into the generative model, we expect to see this discrepancy decrease.

## 5 Conclusions and future work

In this paper we have shown how the SBM can be extended to obtain the MSBM, and presented a principled probabilistic model of images of objects that exploits the MSBM as its model for part shapes. We demonstrated how object segmentations can be obtained simply by performing MCMC inference in the model. The model can also be treated as a probabilistic *evaluator* of segmentations: given a proposal segmentation it can be used to estimate its likelihood. This leads us to believe that the combination of a generative model such as ours, with a discriminative, bottom-up segmentation algorithm could be highly effective. We are currently investigating how textured appearance models, which take into account the spatial structure of pixels, affect the learning and inference algorithms and the performance of the model.

**Acknowledgments**

Thanks to Charless Fowlkes and Vittorio Ferrari for access to datasets, and to Pushmeet Kohli and John Winn for valuable discussions. AE has received funding from the Carnegie Trust, the SORSAS scheme, and the IST Programme under the PASCAL2 Network of Excellence (IST-2007-216886).

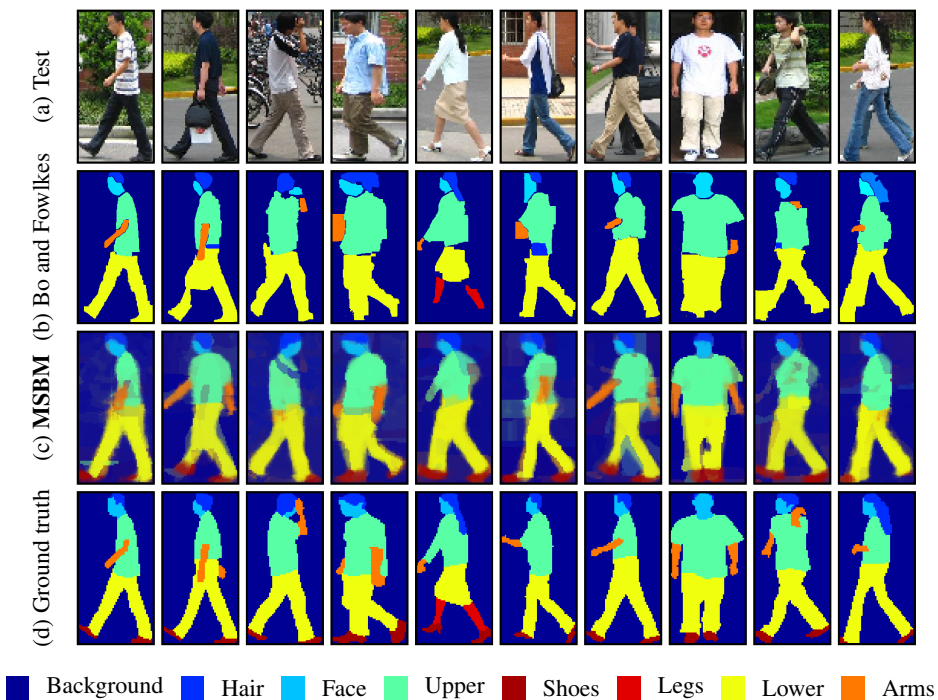

Figure 6: **Penn-Fudan pedestrians.** (a) Test images. (b) Results reported by Bo and Fowlkes [21]. (c) Output of the joint model. (d) Ground-truth images. Images shown are those selected by [21].

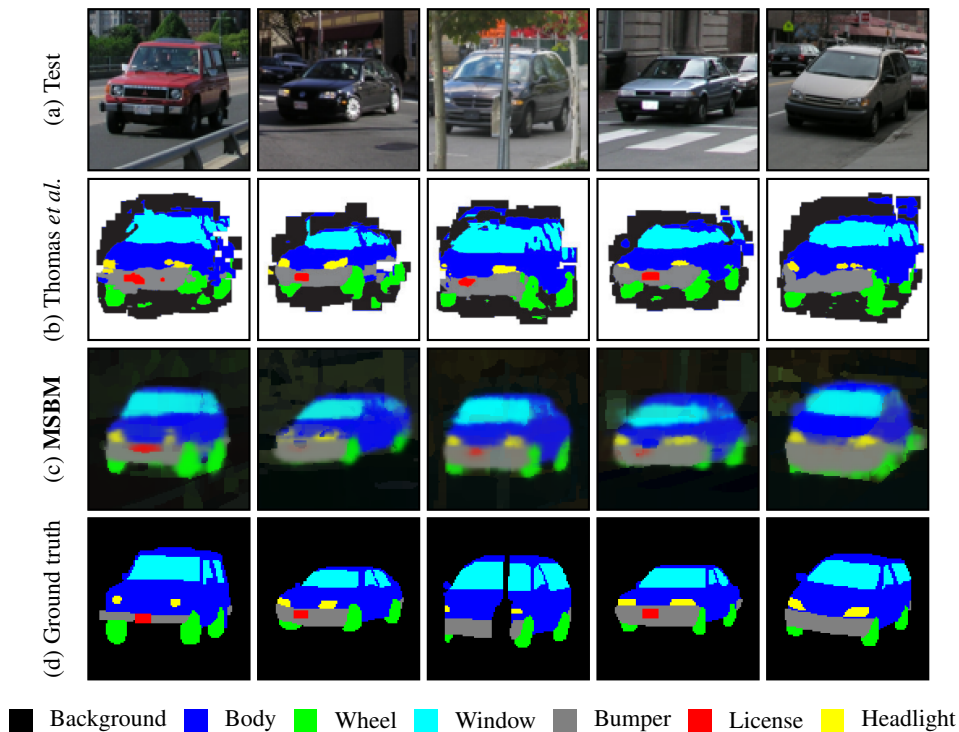

Figure 7: **ETHZ cars.** (a) Test images. (b) Results reported by Thomas *et al.* [22]. (c) Output of the joint model. (d) Ground-truth images. Images shown are those selected by [22].

## Footnotes

[1]We obtained the best quantitative results with these settings. The appearances exhibited by the parts in the dataset are highly varied, and the complexity of the appearance model reflects this fact.

# References

[1] S. M. Ali Eslami, Nicolas Heess, and John Winn. The Shape Boltzmann Machine: a Strong Model of Object Shape. In *IEEE CVPR*, 2012.

[2] Mark Everingham, Luc Van Gool, Christopher K. I. Williams, John Winn, and Andrew Zisserman. The PASCAL Visual Object Classes (VOC) Challenge. *International Journal of Computer Vision*, 88:303–338, 2010.

[3] Martin Fischler and Robert Elschlager. The Representation and Matching of Pictorial Structures. *IEEE Transactions on Computers*, 22(1):67–92, 1973.

[4] David Marr. *Vision: A Computational Investigation into the Human Representation and Processing of Visual Information*. Freeman, 1982.

[5] Irving Biederman. Recognition-by-components: A theory of human image understanding. *Psychological Review*, 94:115–147, 1987.

[6] Ashish Kapoor and John Winn. Located Hidden Random Fields: Learning Discriminative Parts for Object Detection. In *ECCV*, pages 302–315, 2006.

[7] John Winn and Jamie Shotton. The Layout Consistent Random Field for Recognizing and Segmenting Partially Occluded Objects. In *IEEE CVPR*, pages 37–44, 2006.

[8] Nebojsa Jojic and Yaron Caspi. Capturing Image Structure with Probabilistic Index Maps. In *IEEE CVPR*, pages 212–219, 2004.

[9] John Winn and Nebojsa Jojic. LOCUS: Learning object classes with unsupervised segmentation. In *ICCV*, pages 756–763, 2005.

[10] Nebojsa Jojic, Alessandro Perina, Marco Cristani, Vittorio Murino, and Brendan Frey. Stel component analysis. In *IEEE CVPR*, pages 2044–2051, 2009.

[11] S. M. Ali Eslami and Christopher K. I. Williams. Factored Shapes and Appearances for Parts-based Object Understanding. In *BMVC*, pages 18.1–18.12, 2011.

[12] Nicolas Heess. *Learning generative models of mid-level structure in natural images*. PhD thesis, University of Edinburgh, 2011.

[13] Ruslan Salakhutdinov and Geoffrey Hinton. Deep Boltzmann Machines. In *AISTATS*, volume 5, pages 448–455, 2009.

[14] Tijmen Tieleman. Training restricted Boltzmann machines using approximations to the likelihood gradient. In *ICML*, pages 1064–1071, 2008.

[15] Carsten Rother, Vladimir Kolmogorov, and Andrew Blake. "GrabCut": interactive foreground extraction using iterated graph cuts. *ACM SIGGRAPH*, 23:309–314, 2004.

[16] Eran Borenstein, Eitan Sharon, and Shimon Ullman. Combining Top-Down and Bottom-Up Segmentation. In *CVPR Workshop on Perceptual Organization in Computer Vision*, 2004.

[17] Himanshu Arora, Nicolas Loeff, David Forsyth, and Narendra Ahuja. Unsupervised Segmentation of Objects using Efficient Learning. *IEEE CVPR*, pages 1–7, 2007.

[18] Bogdan Alexe, Thomas Deselaers, and Vittorio Ferrari. ClassCut for unsupervised class segmentation. In *ECCV*, pages 380–393, 2010.

[19] Nicolas Heess, Nicolas Le Roux, and John Winn. Weakly Supervised Learning of Foreground-Background Segmentation using Masked RBMs. In *ICANN*, 2011.

[20] Nicolas Le Roux, Nicolas Heess, Jamie Shotton, and John Winn. Learning a Generative Model of Images by Factoring Appearance and Shape. *Neural Computation*, 23(3):593–650, 2011.

[21] Yihang Bo and Charless Fowlkes. Shape-based Pedestrian Parsing. In *IEEE CVPR*, 2011.

[22] Alexander Thomas, Vittorio Ferrari, Bastian Leibe, Tinne Tuytelaars, and Luc Van Gool. Using Recognition and Annotation to Guide a Robot's Attention. *IJRR*, 28(8):976–998, 2009.

[23] Bryan Russell, Antonio Torralba, Kevin Murphy, and William Freeman. LabelMe: A Database and Tool for Image Annotation. *International Journal of Computer Vision*, 77:157–173, 2008.

[24] Leonid Sigal, Alexandru Balan, and Michael Black. HumanEva. *International Journal of Computer Vision*, 87(1-2):4–27, 2010.

[25] Pablo Arbelaez, Michael Maire, Charless C. Fowlkes, and Jitendra Malik. From Contours to Regions: An Empirical Evaluation. In *IEEE CVPR*, 2009.

